# Short-Term Depression in VLSI Stochastic Synapse

**Peng Xu, Timothy K. Horiuchi, and Pamela Abshire**
Department of Electrical and Computer Engineering, Institute for Systems Research
University of Maryland, College Park, MD 20742
`pxu,timmer,pabshire@umd.edu`

## Abstract

We report a compact realization of short-term depression (STD) in a VLSI stochastic synapse. The behavior of the circuit is based on a subtractive single release model of STD. Experimental results agree well with simulation and exhibit expected STD behavior: the transmitted spike train has negative autocorrelation and lower power spectral density at low frequencies which can remove redundancy in the input spike train, and the mean transmission probability is inversely proportional to the input spike rate which has been suggested as an automatic gain control mechanism in neural systems. The dynamic stochastic synapse could potentially be a powerful addition to existing deterministic VLSI spiking neural systems.

## 1 Introduction

Synapses are the primary locations in neural systems where information is processed and transmitted. Synaptic transmission is a stochastic process by nature, *i.e.* it has been observed that at central synapses transmission proceeds in an all-or-none fashion with a certain probability. The synaptic weight has been modeled as $R = npq$ [1], where $n$ is the number of quantal release sites, $p$ is the probability of release per site, and $q$ is some measure of the postsynaptic effect. The synapse undergoes constant changes in order to learn from and adapt to the ever-changing outside world. The variety of synaptic plasticities differ in the triggering condition, time span, and involvement of pre- and postsynaptic activity. Regulation of the vesicle release probability has been considered as the underlying mechanism for various synaptic plasticities [1–3]. The stochastic nature of the neural computation has been investigated and the benefits of stochastic computation such as energy efficiency, communication efficiency, and computational efficiency have been shown [4–6]. Recently there is increasing interest in probabilistic modeling of brain functions [7]. VLSI stochastic synapse could provide a useful hardware tool to investigate stochastic nature of the synapse and also function as the basic computing unit for VLSI implementation of stochastic neural computation.

Although adaptive deterministic VLSI synapses have been extensively studied and developed for neurally inspired VLSI learning systems [8–13], stochastic synapses have been difficult to implement in VLSI because it is hard to properly harness the probabilistic behavior, normally provided by noise. Although stochastic behavior in integrated circuits has been investigated in the context of random number generators (RNGs) [14], these circuits either are too complicated to use for a stochastic synapse or suffer from poor randomness. Therefore other approaches were explored to bring randomness into the systems. Stochastic transmission was implemented in software using a lookup table and a pseudo random number generator [15]. Stochastic transition between potentiation and depression has been demonstrated in bistable synapses driven by stochastic spiking behavior at the network level for stochastic learning [16].

Previously we reported the first VLSI stochastic synapse. Experimental results demonstrated true randomness as well as the adjustable transmission probability. The implementation with $\sim 15$ transistors is compact for these added features, although there are much more compact deterministic

synapses with as few as five transistors. We also proposed the method to implement plasticity and demonstrated the implementation of STD by modulating the *probability* of spike transmission. Like its deterministic counterpart, this stochastic synapse operates on individual spike train inputs; its stochastic character, however, creates the possibility of a broader range of computational primitives such as rate normalization of Poisson spike trains, probabilistic multiplication, or coincidence detection. In this paper we extend the subtractive single release model of STD to the VLSI stochastic synapse. We present the simulation of the new model. We describe a novel compact VLSI implementation of a stochastic synapse with STD and demonstrate extensive experimental results showing the agreement with both simulation and theory over a range of conditions and biases.

## 2 VLSI Stochastic Synapse and Plasticity

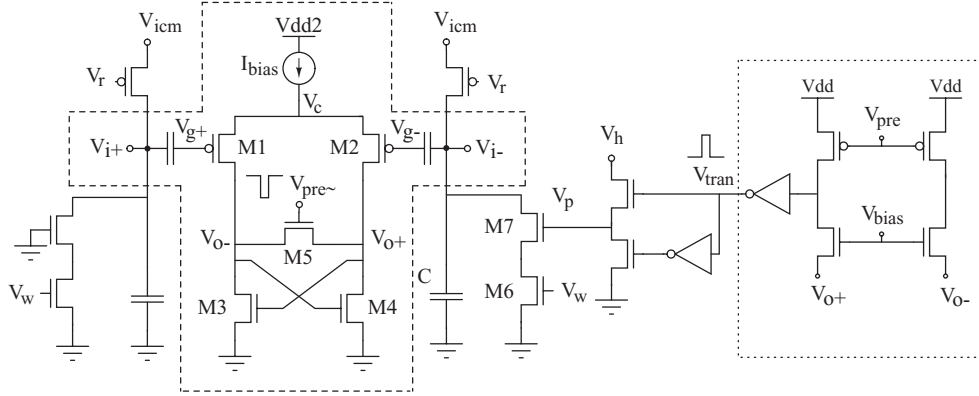

Figure 1: Schematic of the stochastic synapse with STD.

Previously we demonstrated a compact stochastic synapse circuit exhibiting true randomness and consuming very little power (10-44 $\mu$W). The core of the structure is a clocked, cross-coupled differential pair comparator with input voltages $V_{i+}$ and $V_{i-}$, as shown in the dashed box in Fig. 1. It uses competition between two intrinsic circuit noise sources to generate random events. The differential design helps to reduce the influence from other noise sources. When a presynaptic spike arrives, $V_{pre\sim}$ goes low, and transistor M5 shuts off. $V_{o+}$ and $V_{o-}$ are nearly equal and the circuit is in its metastable state. When the two sides are closely matched, the imbalance between $V_{o+}$ and $V_{o-}$ caused by current noise in M1-M4 eventually triggers positive feedback, which drives one output to $V_c$ and the other close to ground. We use a dynamic buffer, shown in the dotted box in Fig. 1, to generate rail-to-rail transmitted spikes $V_{tran}$. $V_{tran}$ either goes high (with probability $p$) or stays low (with probability $1 - p$) during an input spike, emulating stochastic transmission.

Fabrication mismatch in an uncompensated stochastic synapse circuit would likely permanently bias the circuit to one solution. In this circuit, floating gate inputs to a pFET differential pair allow the mismatch to be compensated. By controlling the common-mode voltage of the floating gates, we operate the circuit such that hot-electron injection occurs only on the side where the output voltage is close to ground. Over multiple clock cycles hot-electron injection works in negative feedback to equalize the floating gate voltages, bringing the circuit into stochastic operation. The procedure can be halted to achieve a specific probability or allowed to reach equilibrium (50% transmission probability).

The transmission probability can be adjusted by changing the input offset or the floating gate charges. The higher $V_{g+}$ is, the lower $p$ is. The probability tuning function is closely fitted by an error function $f(v) = 0.5 \left(1 + \text{erf}\left(\frac{v-\mu}{\sqrt{2}\delta}\right)\right)$, where $\mu$ is the input offset voltage for $p = 50\%$, $\delta$ is the standard deviation characterizing the spread of the probability tuning, and $v = V_{i-} - V_{i+}$ is the input offset voltage. Synaptic plasticity can be implemented by dynamically modulating the probability. Input offset modulation is suitable for short-term plasticity. Short-term depression is triggered by the transmitted input spikes $V_{tran}$ to emulate the probability decrease because of vesicle depletion. Short-term facilitation is triggered by the input spikes $V_{pre}$ to emulate the probability

increase because of presynaptic $Ca^{2+}$ accumulation. Nonvolatile storage at the floating gate is suitable for long-term plasticity. STDP can be implemented by modulating the probability depending on the precise timing relation between the pre- and postsynaptic spikes.

## 3 Short-Term Depression: Model and Simulation

Although long-term plasticity has attracted much attention because of its apparent association with learning and memory, the functional role of short-term plasticity has only recently begun to be understood. Recent evidence suggests that short-term synaptic plasticity is involved in many functions such as gain control [17], phase shift [18], coincidence detection, and network reconfiguration [19]. It has also been shown that depressing stochastic synapses can increase information transmission efficiency by filtering out redundancy in presynaptic spike trains [5].

Activity dependent short-term changes in synaptic efficacy at the macroscopic level are determined by activity dependent changes in vesicle release probability at the microscopic level. We will focus on STD here. STD during repetitive stimulation results from a decrease in released vesicles. Since there is a finite pool of vesicles, and released vesicles cannot be replenished immediately, a successful release triggered by one spike potentially reduces the probability of release triggered by the next spike. We propose an STD model based on our VLSI stochastic synapse that closely emulates the simple subtractive single release model [5, 20]. A presynaptic spike that is transmitted reduces the input offset voltage $v$ at the VLSI stochastic synapse by $\Delta v$, so that the transmission probability $p(t)$ is reduced. Between successful releases, $v$ relaxes back to its maximum value $v_{max}$ exponentially with a time constant $\tau_d$ so that $p(t)$ relaxes back to its maximum value $p_{max}$ as well. The model can be written as

$$v(t_+) = v(t_-) - \Delta v, \text{ successful transmission at } t \tag{1}$$

$$\tau_d \frac{dv(t)}{dt} = v_{max} - v(t) \tag{2}$$

$$p(t) = f(v(t)) \tag{3}$$

For an input spike train with Poisson arrivals, the model can be expressed as a stochastic differential equation

$$dv = \frac{v_{max} - v}{\tau_d} dt - \Delta v \cdot dN_{p \cdot r(t)} \tag{4}$$

where $dN_{p \cdot r(t)}$ is a Poisson counting process with rate $p \cdot r(t)$, and $r(t)$ is the input spike rate. By taking the expectation $E(\cdot)$ on both sides, we obtain a differential equation

$$\frac{dE(v)}{dt} = \frac{v_{max} - E(v)}{\tau_d} - \Delta v \cdot E(p) r(t) \tag{5}$$

When $v$ is reduced, the probability that it will be reduced again becomes smaller. $v$ is effectively constrained to a small range where we can approximate the function $f(v) = 0.5 \left(1 + \mathtt{erf}\left(\frac{v-\mu}{\sqrt{2}\delta}\right)\right)$ by a linear function $f(v) = av + 0.5$, where $\mu = 0$ for simplicity. We can then solve for $E(p)$ at steady state:

$$p_{ss} \approx \frac{av_{max} + 0.5}{1 + a\Delta v \tau_d r} \approx \frac{p_{max}}{a\Delta v \tau_d r} \propto \frac{1}{r} \tag{6}$$

Therefore the steady state mean probability is inversely proportional to the input spike rate when $a\Delta v \tau_d r \gg 1$. This is consistent with prior work that modeled STD at the macroscopic level [17].

We simulated the model (1)-(3). We use the function $f(v) = 0.5 \left(1 + \mathtt{erf}\left(\frac{v}{\sqrt{2} \cdot 2.16}\right)\right)$, obtained from the best fit of the experimental data. Initially $v$ is set to 5 mV which sets $p_{max}$ close to 1. Although the transformation from $v$ to $p$ is nonlinear, both simulation and experimental data show that this implementation exhibits behavior similar to the model with the linear approximation and the biological data. Fig. 2(a) and 2(b) show that the mean probability is a linear function of the inverse of the input spike rate at various $\Delta v$ and $\tau_d$ for high input spike rates. Both $\Delta v$ and $\tau_d$ affect the slope of the linear relation, following the trend suggested by (6): the bigger the $\Delta v$ or the bigger the $\tau_d$, the smaller the slope is. Fig. 3 shows a simulation of the transient probability for a period of 200 ms. Fig. 4 shows that the output spike train exhibits negative autocorrelation at small time intervals and lower power spectral density (PSD) at low frequencies. This is a direct consequence of STD.

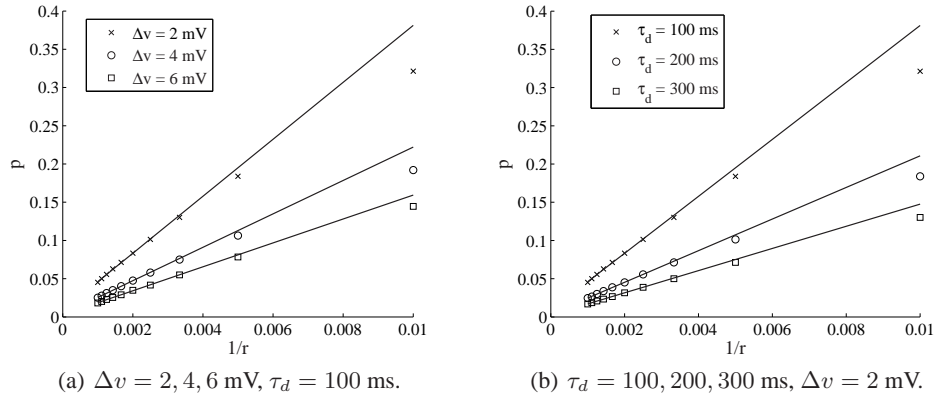

(a) $\Delta v = 2, 4, 6$ mV, $\tau_d = 100$ ms.　　　(b) $\tau_d = 100, 200, 300$ ms, $\Delta v = 2$ mV.

Figure 2: Mean probability as a function of input spike rate from simulation. Data were collected at input rates from 100 Hz to 1000 Hz at 100 Hz intervals. The solid lines show the least mean square fit for input rates from 400 Hz to 1000 Hz.

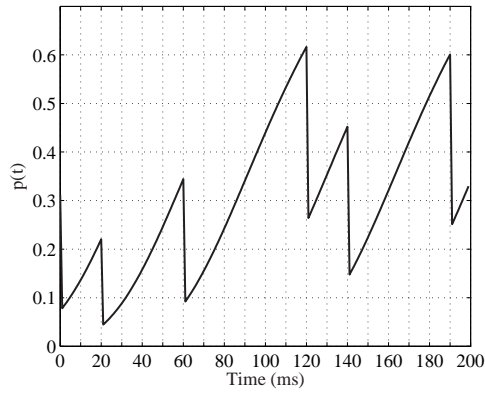

Figure 3: Simulated probability trajectory over 200 ms period. $r = 100$ Hz, $\tau = 100$ ms, $\Delta v = 2$ mV.

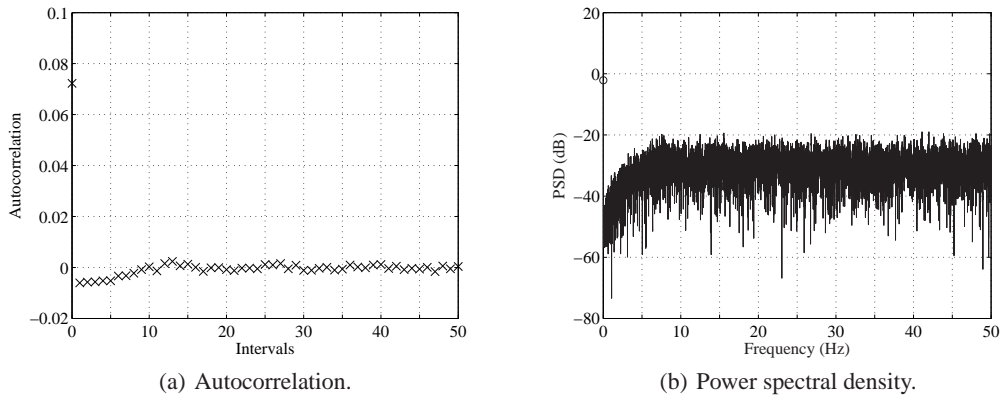

(a) Autocorrelation.　　　(b) Power spectral density.

Figure 4: Characterization of the output spike train from the simulation of the stochastic synapse with STD. $r = 100$ Hz, $\tau_d = 200$ ms, $\Delta v = 6$ mV, $V_{max} = 5$ mV.

# 4 VLSI Implementation of Short-Term Depression

We implemented this model using the stochastic synapse circuit described above (see Fig. 1). Both inputs are restored up to an equilibrium value $V_{icm}$ by tunable resistors implemented by subthreshold pFETs operating in the ohmic region. To change the transmission probability we only need to modulate one side of the input, in this case $V_{i-}$. The resistor and capacitor provide for exponential recovery of the voltage to its equilibrium value. The input $V_{i-}$ is modulated by transistors M6 and M7 based on the result of the previous spike transmission. Every time a spike is transmitted successfully, a pulse with height $V_h$ and width $T_p$ is generated at $V_p$. $T_p$ is same as the input spike pulse width. This pulse discharges the capacitor with a small current determined by $V_w$ and reduces $V_{i-}$ by a small amount, thus decreasing the transmission probability. The value of the tunable resistors is controlled by the gate voltage of the pFETs, $V_r$. When $V_{i-}$ is reduced, the probability that it will be reduced again becomes smaller. Since the probability tuning only occurs in a small voltage range ($\sim$ 10 mV), the change in $V_{i-}$ is limited to this small range as well. Under this special condition, the resistance implemented by the subthreshold pFET is linear and large ($\sim$ G$\Omega$). With capacitance as small as 100 fF, the exponential time constant is tens of milliseconds and is adjustable. Similar control circuits can be applied to $V_{i+}$ to implement short-term facilitation. The update mechanism would then be driven by the presynaptic spike rather than the successfully transmitted spike. The extra components on the left provide for future implementation of short-term facilitation and also symmetrize the stochastic synapse, improving its randomness.

# 5 Experimental Results

The circuit has been fabricated in a commercially-available 0.5 $\mu$m CMOS process with 2 polysilicon layers and 3 metal layers. The layout size of the stochastic synapse is 151.9 $\mu$m $\times$ 91.7 $\mu$m and the layout size of the STD block is 35 $\mu$m $\times$ 32.2 $\mu$m. A 2-to-1 multiplexer with size 35 $\mu$m $\times$ 30 $\mu$m is used to enable or disable STD. As a proof of concept, the layout of the circuit is quite conservative. Assuming no loss of performance, the existing circuit area could be reduced by 50%.

The circuit uses a nominal power supply of 5 V for normal operation. The differential pair comparator uses a separate power supply for hot-electron injection. Each floating-gate pFET has a tunnelling structure, which is a source-drain connected pFET with its gate connected to the floating node. A separate power supply provides the tunnelling voltage to the shorted source and drain (tunnelling node). When the tunnelling voltage is high enough ($\sim$14-15 V), electron tunnels through the silicon dioxide, from the floating gate to the tunnelling node. We use this phenomenon to remove electrons from the floating gate only during initialization. Alternatively Ultra-Violet (UV) activated conductances may be used to remove electrons from the gate to avoid the need for special power supplies.

To begin the test, we first remove residual charges on the floating gates in the stochastic synapse. We set $V_{icm}$ = 2 V. We raise the power supply of the differential pair comparator to 5.3 V to facilitate the hot-electron injection. We use the negative feedback operation of hot-electron injection described above to automatically program the circuit into its stochastic regime. We halt the injection by lowering the power supply to 5 V. During this procedure, STD is disabled, so that the probability at this operating point is the synaptic transmission probability without any dynamics.

We then enable STD. We use a signal generator to generate pulse signals which serve as input spikes. Although spike trains are better modeled by Poisson arrivals, the averaging behavior should be similar for deterministic spike trains which make testing easier. We use $I_{bias}$ = 100 nA. The power consumption of the STD block is much smaller than the stochastic synapse. The total power consumption is about 10 $\mu$W.

We collect output spikes from the depressing stochastic synapse at an input spike rate of 100 Hz. We divide time into bins according to the input spike rate so that in each bin there is either 1 or 0 output spike. In this way, we convert the output spike train into a bit sequence s(k). We then compute the normalized autocorrelation, defined as $A(n) = E(s(k)s(k+n)) - E^2(s(k))$, where $n$ is the number of time intervals between two bits. $A(0)$ gives the variance of the sequence. For two bits with distance $n > 0$, $A(n) = 0$ if they are independent, indicating good randomness, and $A(n) < 0$ if they are anticorrelated, indicating the depressing effect of preceding spikes on the later spikes. Fig. 5 shows the autocorrelation of the output spike trains at two different $V_r$. There is significant nega-

tive correlation at small time intervals and little correlation at large time intervals, as expected from STD. Fig. 6 shows the PSD of the output spike trains from the same data shown in Fig. 5. Clearly, the PSD is reduced at low frequencies. The time constant of STD increases with $V_r$ so that the larger $V_r$ is, the longer the period of the negative autocorrelation is and the lower the frequencies where power is reduced. This agrees with simulation results. Notice that the autocorrelation and PSD for $V_r = 1.59$ V show very close similarity to the simulation results in Fig. 4. Normally redundant information is represented by positive autocorrelation in the time domain, which is characterized by power at low frequencies. By reducing the low frequency component of the spike train, redundant information is suppressed and overall information transmission efficiency is improved. If the negative autocorrelation of the synaptic dynamics matches the positive autocorrelation in the input spike train, the redundancy is cancelled and the output is uncorrelated [5].

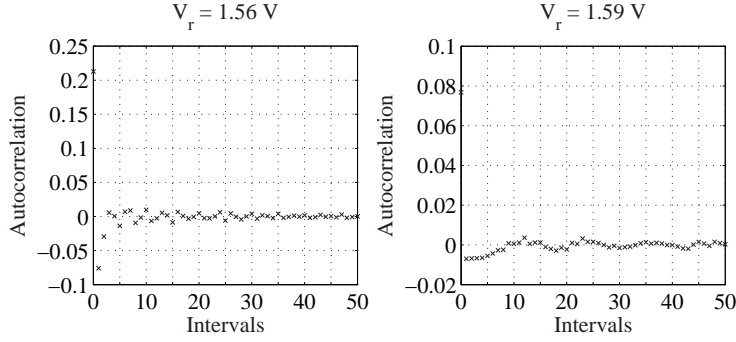

Figure 5: Autocorrelation of output spike trains from the VLSI stochastic synapse with STD for an input spike rate of 100 Hz. Autocorrelation at zero time represents the sequence variance, and negative autocorrelation at short time intervals indicates STD.

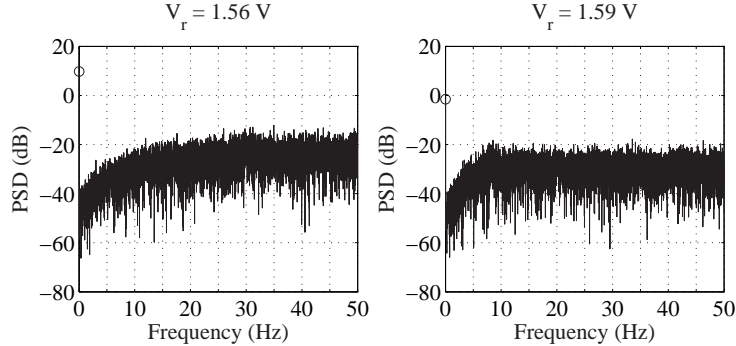

Figure 6: Power spectral density of output spike trains from the VLSI stochastic synapse with STD for an input spike rate of 100 Hz. Lower PSD at low frequencies indicates STD.

We collect output spikes in response to $10^4$ input spikes at input spike rates from 100 Hz to 1000 Hz with 100 Hz intervals. Fig. 7(a) shows that the mean transmission probability is inversely proportional to the input spike rate for various pulse widths when the rate is high enough. This matches the theoretical prediction in (6) very well. By scaling the probability with the input spike rate, the synapse tends to normalize the DC component of input frequency and preserve the neuron dynamic range, thus avoiding saturation due to fast firing presynaptic neurons and retaining sensitivity to less frequently firing neurons [17]. The slope of mean probability decreases as the pulse width increases. Since the pulse width determines the discharging time of the capacitor at $V_{i-}$, the larger the pulse width, the larger the $\Delta v$ is and the smaller the slope is. Fig. 7(b) shows that $a\Delta v\tau_d$ scales linearly with the pulse width. The discharging current is approximately constant, thus $\Delta v$ is proportional to the pulse width.

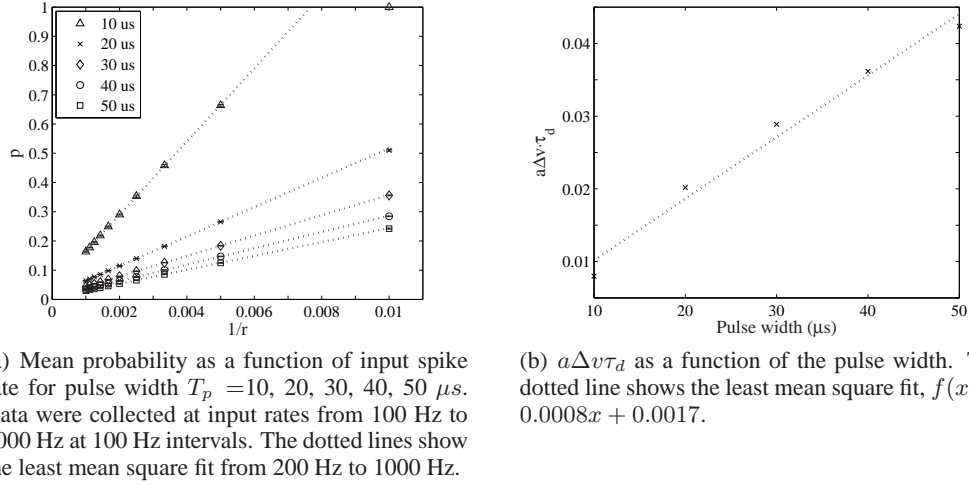

(a) Mean probability as a function of input spike rate for pulse width $T_p$ =10, 20, 30, 40, 50 $\mu s$. Data were collected at input rates from 100 Hz to 1000 Hz at 100 Hz intervals. The dotted lines show the least mean square fit from 200 Hz to 1000 Hz.

(b) $a\Delta v\tau_d$ as a function of the pulse width. The dotted line shows the least mean square fit, $f(x) = 0.0008x + 0.0017$.

Figure 7: Steady state behavior of VLSI stochastic synapse with STD for different pulse widths.

We perform the same experiments for different $V_r$ and $V_w$. As $V_r$ increases, the slope of mean transmission probability as a linear function of $\frac{1}{r}$ decreases. This is due to the increasing $\tau_d = RC$, where the equivalent resistance $R$ from the pFET increases with $V_r$. Fig. 8(a) shows that $a\Delta v\tau_d$ is approximately an exponential function of $V_r$, indicating that the equivalent $R$ of the pFET is approximately exponential to its gate voltage $V_r$. For $V_w$, the slope of mean transmission probability decreases as $V_w$ increases. This is due to the increasing $\Delta v$ with $V_w$. Fig. 8(b) shows that $a\Delta v\tau_d$ is approximately an exponential function of $V_w$, indicating that the discharging current from the transistor M6 is approximately exponential to its gate voltage $V_w$. This matches the I-V characteristics of the MOSFET in subthreshold.

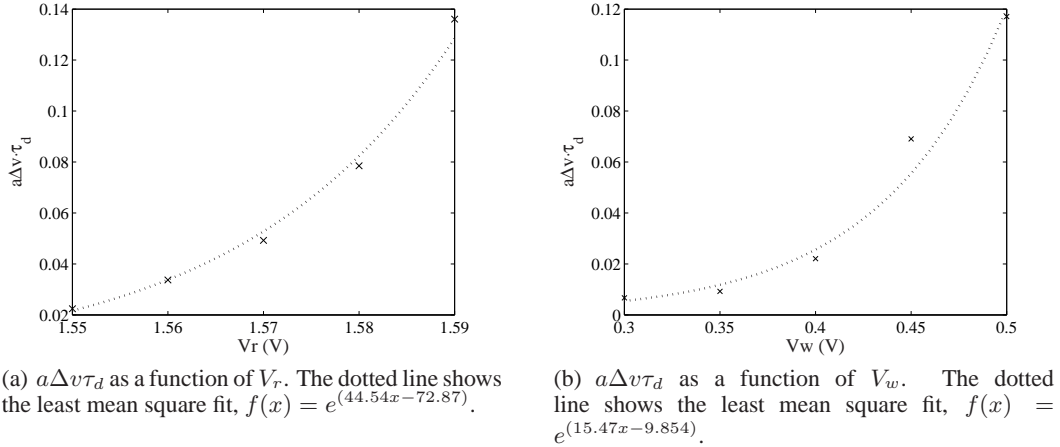

(a) $a\Delta v\tau_d$ as a function of $V_r$. The dotted line shows the least mean square fit, $f(x) = e^{(44.54x-72.87)}$.

(b) $a\Delta v\tau_d$ as a function of $V_w$. The dotted line shows the least mean square fit, $f(x) = e^{(15.47x-9.854)}$.

Figure 8: The effect of biases $V_r$ and $V_w$ on the depressing behavior.

## 6 Conclusion

We designed and tested a VLSI stochastic synapse with short-term depression. The behavior of the depressing synapse agrees with theoretical predictions and simulation. The strength and time duration of the depression can be tuned by the biases. The circuit is compact and consumes low power. It is a good candidate to bring randomness and rich dynamics into VLSI spiking neural

systems, such as for rate-independent coincidence detection of Poisson spike trains. However, the application of such dynamic stochastic synapses in large networks still remains a challenge.

## References

[1] C. Koch, *Biophysics of Computation: Information Processing in Single Neurons.* New York, NY: Oxford University Press, 1999.

[2] M. V. Tsodyks and H. Markram, "The neural code between neocortical pyramidal neurons depends on neurotransmitter release probability," *Proc. Natl. Acad. Sci. USA*, vol. 94, pp. 719–723, 1997.

[3] W. Senn, H. Markram, and M. Tsodyks, "An algorithm for modifying neurotransmitter release probability based on pre- and postsynaptic spike timing," *Neural Computation*, vol. 13, pp. 35–67, 2000.

[4] W. Maass and A. M. Zador, "Dynamic stochastic synapses as computational units," *Neural Comput.*, vol. 11, no. 4, pp. 903–917, 1999.

[5] M. S. Goldman, P. Maldonado, and L. F. Abbott, "Redundancy reduction and sustained firing with stochastic depressing synapses," *J. Neurosci.*, vol. 22, no. 2, pp. 584–591, 2002.

[6] W. B. Levy and R. A. Baxter, "Energy-efficient neuronal computation via quantal synaptic failures," *J. Neurosci.*, vol. 22, no. 11, pp. 4746–4755.

[7] R. Rao, B. Olshausen, and M. Lewicki, Eds., *Statistical Theories of the Brain.* MIT Press, 2001.

[8] C. Diorio, P. Hasler, B. A. Minch, and C. Mead, "A single-transistor silicon synapse," *IEEE Trans. Electron Devices*, vol. 43, pp. 1972–1980, Nov. 1996.

[9] P. Häfliger and M. Mahowald, "Spike based normalizing Hebbian learning in an analog VLSI artificial neuron," *Int. J. Analog Integr. Circuits Signal Process.*, vol. 18, no. 2-3, pp. 133–139, 1999.

[10] S.-C. Liu, "Analog VLSI circuits for short-term dynamic synapses," *EURASIP Journal on Applied Signal Processing*, vol. 2003, pp. 620–628, 2003.

[11] E. Chicca, G. Indiveri, and R. Douglas, "An adaptive silicon synapse," in *Proc. IEEE Int. Symp. Circuits Systems*, vol. 1, Bangkok, Thailand, May 2003, pp. 81–84.

[12] A. Bofill, A. F. Murray, and D. P. Thompson, "Circuits for VLSI implementation of temporally asymmetric Hebbian learning," in *Advances in Neural Information Processing Systems*, S. B. T. G. Dietterich and Z. Ghahramani, Eds. Cambridge, MA, USA: MIT Press, 2002.

[13] G. Indiveri, E. Chicca, and R. Douglas, "A VLSI array of low-power spiking neurons and bistable synapses with spike-timing dependent plasticity," *IEEE Trans. Neural Networks*, vol. 17, pp. 211–221, 2006.

[14] C. S. Petrie and J. A. Connelly, "A noise-based IC random number generator for applications in cryptography," *IEEE Trans. Circuits Syst. I*, vol. 47, no. 5, pp. 615–621, May 2000.

[15] D. H. Goldberg, G. Cauwenberghs, and A. G. Andreou, "Probabilistic synaptic weighting in a reconfigurable network of VLSI integrate-and-fire neurons," *Neural Networks*, vol. 14, pp. 781–793, 2001.

[16] S. Fusi, M. Annunziato, D. Badoni, A. Salamon, and D. J. Amit, "Spike driven synaptic plasticity: theory, simulation, VLSI implementation," *Neural Computation*, vol. 12, pp. 2227–2258, 2000.

[17] L. F. Abbott, J. A. Varela, K. Sen, and S. B. Nelson, "Synaptic depression and cortical gain control," *Science*, vol. 275, pp. 220–224, 1997.

[18] F. S. Chance, S. B. Nelson, and L. F. Abbott, "Synaptic depression and the temporal response characteristics of V1 cells," *J. Neurosci.*, vol. 18, no. 12, pp. 4785–4799, 1998.

[19] F. Nadim and Y. Manor, "The role of short-term synaptic dynamics in motor control," *Curr. Opin. Neurobiol.*, vol. 10, pp. 683–690, Dec. 2000.

[20] R. S. Zucker, "Short-term synaptic plasticity," *Ann. Rev. Neurosci.*, vol. 12, pp. 13–31, 1989.
